# DYNAMICS OF ANALOG NEURAL NETWORKS WITH TIME DELAY

C.M. Marcus and R.M. Westervelt
Division of Applied Sciences and Department of Physics
Harvard University, Cambridge Massachusetts 02138

## ABSTRACT

A time delay in the response of the neurons in a network can induce sustained oscillation and chaos. We present a stability criterion based on local stability analysis to prevent sustained oscillation in symmetric delay networks, and show an example of chaotic dynamics in a non-symmetric delay network.

## I. INTRODUCTION

Understanding how time delay affects the dynamics of neural networks is important for two reasons: First, some degree of time delay is intrinsic to any physically realized network, both in biological neural systems and in electronic artificial neural networks. As we will show, it is not obvious what constitutes a "small" (i.e. ignorable) delay which will not qualitatively change the network dynamics. For some network configurations, delay much smaller than the intrinsic relaxation time of the network can induce collective oscillatory behavior not predicted by mathematical models which ignore delay. These oscillations may or may not be desirable; in either case, one should understand when and how new dynamics can appear. The second reason to study time delay is for its intentional use in parallel computation. The dynamics of neural networks which always converge to fixed points are now fairly well understood. Several neural network models have appeared recently which use time delay to produce dynamic computation such as associative recall of sequences [Kleinfeld,1986; Sompolinsky and Kanter,1986]. It has also been suggested that time delay produces an effective noise in the network dynamics which can yield improved recall of memories [Conwell, 1987] Finally, to the extent that neural networks research is inspired by biological systems, the known presence of time delays in a many real neural systems suggests their usefulness in parallel computation.

In this paper we will show how time delay in an analog neural network can produce sustained oscillation and chaos. In section 2 we consider the case of a symmetrically connected network. It is known [Cohen and Grossberg,1983; Hopfield, 1984] that in the absence of time delay a symmetric network will always converge to a fixed point attractor. We show that adding a fixed delay to the response of each neuron will produce sustained oscillation when the magnitude of the delay exceeds a critical value, which depends on the neuron gain and the network connection topology. We then analyze the

all-inhibitory and symmetric ring topologies as examples. In section 3, we discuss chaotic dynamics in asymmetric neural networks, and give an example of a small (N=3) network which shows delay-induced chaos. The analytical results presented here are supported by numerical simulations and experiments performed on a small electronic neural network with controllable time. A detailed derivation of the stability results for the symmetric network is given in [Marcus and Westervelt, 1989], and the electronic circuit used is described in described [Marcus and Westervelt, 1988].

## II. STABILITY OF SYMMETRIC NETWORKS WITH DELAY

The dynamical system we consider describes an electronic circuit of N saturable amplifiers ("neurons") coupled by a resistive interconnection matrix. The neurons do not respond to an input voltage $u_i$ instantaneously, but produce an output after a delay, which we take to be the same for all neurons. The neuron input voltages evolve according to the following equations:

$$\dot{u}_i(t) = -u_i(t) + \sum_{j=1}^{N} J_{ij} f(u_j(t-\tau)). \tag{1}$$

The transfer function for each neuron is taken to be an identical sigmoidal function f(u) with a maximum slope $df/du \equiv \beta$ at $u = 0$. The unit of time in these equations has been scaled to the characteristic network relaxation time, thus $\tau$ can be thought of as the ratio of delay time to relaxation time. The symmetric interconnection matrix $J_{ij}$ describes the conductance between neurons i and j is normalized to satisfy $\Sigma_j |J_{ij}| = 1$ for all i. This normalization assumes that each neuron sees the same conductance at its input [Marcus and Westervelt, 1989]. The initial conditions for this system are a set of N continuous functions defined on the interval $-\tau \leq t \leq 0$. We take each initial function to be constant over that interval, though possibly different for different i. We find numerically that the results do not depend on the form of the initial functions.

### Linear Stability Analysis at Low Gain

Studying the stability of the fixed point at the origin ($u_i = 0$ for all i) is useful for understanding the source of delay-induced sustained oscillation and will lead to a low-gain stability criterion for symmetric networks. It is important to realize however, that for the system (1) with a sigmoidal nonlinearity, if the origin is stable then it is the unique attractor, which makes for rather uninteresting dynamics. Thus the origin will almost certainly be unstable in any useful configuration. Linear stability analysis about the origin will show that at $\tau = 0$, as the gain $\beta$ is increased, the origin always loses stability by a type of bifurcation which only produces other fixed points, but for $\tau > 0$ an alternative type of bifurcation of the origin can occur which produces the sustained oscillatory modes. The stability criterion derived insures that this alternate bifurcation - a Hopf bifurcation - does not occur.

The natural coordinate system for the linearized version of (1) is the set of N eigenvectors of the connection matrix $J_{ij}$, defined as $x_i(t)$, i=1,..N. In terms of the $x_i(t)$,

the linearized system can be written

$$\dot{x}_i(t) = -x_i(t) + \beta \lambda_i\, x_i(t-\tau) \tag{2}$$

where $\beta$ is the neuron gain and $\lambda_i$ (i=1,..N) are the eigenvalues of $J_{ij}$. In general, these eigenvalues have both real and imaginary parts; for $J_{ij} = J_{ji}$ the $\lambda_i$ are purely real. Assuming exponential time evolution of the form $x_i(t) = x_i(0)e^{s_i t}$, where $s_i$ is a complex characteristic exponent, yields a set of N transcendental characteristic equations: $(s_i + 1)e^{s_i \tau} = \beta \lambda_i$. The condition for stability of the origin, $Re(s_i) < 0$ for all i, and the characteristic equations can be used to specify a stability region in the complex plane of eigenvalues, as illustrated in Fig. (1a). When all eigenvalues of $J_{ij}$ are within the stability region, the origin is stable.  For $\tau = 0$, the stability region is defined by $Re(\lambda) < 1/\beta$, giving a half-plane stability condition familiar from ordinary differential equations. For $\tau > 0$, we define the border of the stability region $\Lambda(\theta)$ at an angle $\theta$ from the $Re(\lambda)$ axis as the radial distance from the point $\lambda = 0$ to the first point (i.e. smallest value of $\Lambda(\theta)$) which satisfies the characteristic equation for purely imaginary characteristic exponent $s_j \equiv i\omega_j$. The delay-dependent value of $\Lambda(\theta)$ is given by

$$\Lambda(\theta) = \frac{1}{\beta}\sqrt{\omega^2 + 1} \quad ; \quad \omega = -\tan(\omega\tau - \theta) \tag{3}$$

where $\omega$ is in the range $(\theta - \pi/2) \leq \omega\tau \leq \theta$, modulo $2\pi$.

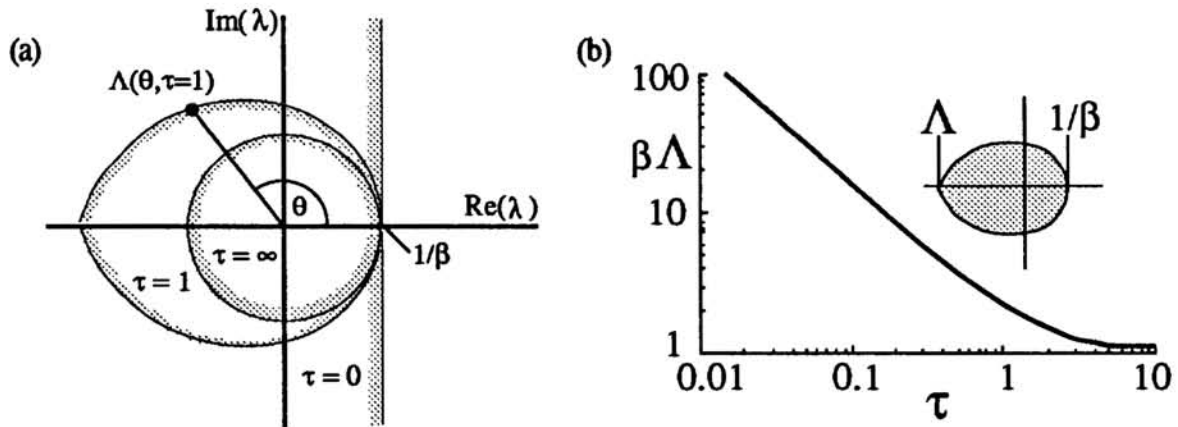

**Figure 1.** (a) Regions of Stability in the Complex Plane of Eigenvalues $\lambda$ of the Connection Matrix $J_{ij}$. for $\tau = 0,1,\infty$. (b) Where Stability Region Crosses the Real-$\lambda$ Axis in the Negative Half Plane.

Notice that for nonzero delay the stability region closes on the $Re(\lambda)$ axis in the negative half-plane. It is therefore possible for *negative* real eigenvalues to induce an instability of the origin. Specifically, if the minimum eigenvalue of the symmetric matrix $J_{ij}$ is more negative than $-\Lambda(\theta = \pi)$ then the origin is unstable. We define this "back door" to the stability region along the real axis as $\Lambda > 0$, dropping the argument $\theta = \pi$. $\Lambda$ is inversely proportional to the gain $\beta$ and depends on delay as shown in Fig. (1b). For large and small delay, $\Lambda$ can be approximated as an explicit function of delay and gain:

$$\Lambda \;\cong\; \begin{cases} (1/\beta)\,\pi/2\tau & \tau << 1 \qquad (4a) \\[4mm] (1/\beta)\!\left[1+(\pi/(\tau+1))^2\right]^{1/2} & \tau >> 1 \qquad (4b) \end{cases}$$

In the infinite-delay limit, the delay-differential system (1) is equivalent to an iterated map or parallel-update network of the form $u_i(t+1) = \Sigma_j\, J_{ij}\, f(u_j(t))$ where t is a discrete iteration index. In this limit, the stability region is circular, corresponding to the fixed point stability condition for the iterated map system.

Consider the stability of the origin in a symmetrically connected delay system (1) as the neuron gain $\beta$ is increased from zero to a large value. A bifurcation of the origin will occur when the maximum eigenvalue $\lambda_{max} > 0$ of $J_{ij}$ becomes larger than $1/\beta$ or when the minimum eigenvalue $\lambda_{min} < 0$ becomes more negative than $-\Lambda = -\beta^{-1}(\omega^2+1)^{1/2}$, where $\omega = -\tan(\omega\tau)$, $[\pi/2 < \omega < \pi]$. Which bifurcation occurs first depends on the delay and the eigenvalues of $J_{ij}$. The bifurcation at $\lambda_{max} = \beta^{-1}$ is a pitchfork (as it is for $\tau = 0$) corresponding to a characteristic exponent $s_i$ crossing into the positive real half plane along the real axis. This bifurcation creates a pair of fixed points along the eigenvector $x_i$ associated with that eigenvalue. These fixed points constitute a single memory state of the network. The bifurcation at $\lambda_{min} = -\Lambda$ corresponds to a Hopf bifurcation [Marsden and McCracken,1976], where a pair of characteristic exponents pass into the real half plane with imaginary components $\pm\omega$ where $\omega = -\tan(\omega\tau)$, $[\pi/2 < \omega < \pi]$. This bifurcation, not present at $\tau = 0$, creates an oscillatory attractor along the eigenvector associated with $\lambda_{min}$.

A simple stability criterion can be constructed by requiring that the most negative eigenvalue of the (symmetric) connection matrix not be more negative than $-\Lambda$. Because $\Lambda$ is always larger than its small-delay limit $\pi/(2\tau\beta)$, the criterion can be stated as a limit on the size on the delay (in units of the network relaxation time.)

$$\tau < -\frac{\pi}{2\beta\lambda_{min}} \qquad \Rightarrow \qquad \text{no sustained oscillation.} \qquad (5)$$

Linear stability analysis does not prove global stability, but the criterion (5) is supported by considerable numerical and experimental evidence [Marcus and Westervelt, 1989]. For long delays, where $\Lambda \cong \beta^{-1}$, linear stability analysis suggests that sustained oscillation will not exist as long as $-\beta^{-1} < \lambda_{min}$. In the infinite-delay limit, it can be shown that this condition insures global stability in the discrete-time parallel-update network. [Marcus and Westervelt, to appear].

At large gain, Eq. (5) does not provide a useful stability criterion because the delay required for stability tends to zero as $\beta \to \infty$. The nonlinearity of the transfer function becomes important at large gain and stable, fixed-point-only dynamics are found at large gain and nonzero delay, indicating that Eq. (5) is overly conservative at large gain. To understand this, we must include the nonlinearity and consider the stability of the oscillatory modes themselves. This is described in the next section.

## Stability in the Large-Gain Limit

We now analyze the oscillatory mode at large gain for the particular case of coherent oscillation. We find a second stability criterion which predicts a gain-independent critical delay below which all initial conditions lead to fixed points. This result complements the low gain result of the previous section for this class of network; experimentally and numerically we find excellent agree in both regimes, with a cross-over at the value of gain where fixed points appear away from the origin, $\beta = 1/\lambda_{max}$.

In considering only coherent oscillation, we not only assume that $J_{ij}$ is symmetric but that its maximum and minimum eigenvalues satisfy $0 < \lambda_{max} < -\lambda_{min}$ and that the eigenvector associated with $\lambda_{min}$ points in a coherent direction, defined to be along any of the $2^N$ vectors of the form $(\pm 1, \pm 1, \pm 1, ...)$ in the $u_i$ basis. For this case, we find that in the limit of infinite gain, where the nonlinearity is of the form $f(u) = sgn(u)$, multiple fixed point attractors coexist with the oscillatory attractor and that the size of the basin of attraction for the oscillatory mode varies with the delay [Marcus and Westervelt, 1988]. At a critical value of delay $\tau_{crit}$ the basin of attraction for oscillation vanishes and the oscillatory mode loses stability. In [Marcus and Westervelt, 1989] we show:

$$\tau_{crit} = -\ln\left(1 + \lambda_{max} / \lambda_{min}\right) \qquad (6)$$

For delays less than this critical value, all initial states lead to stable fixed points.

Notice that the critical delay for coherent oscillation diverges as $|\lambda_{max}/\lambda_{min}| \to 1^-$. Experimentally and numerically we find that this prediction has more general applicability: None of the symmetric networks investigated which satisfied $|\lambda_{max}/\lambda_{min}| \geq 1$ (and $\lambda_{max} > 0$) showed sustained oscillation for $\tau < \sim 10$. This observation is a useful criterion for electronic circuit design, where single-device delays are generally shorter than the circuit relaxation time ($\tau < 1$), but only the case of coherent oscillation is supported by analysis.

## Examples

As a first example, we consider the fully-connected all-inhibitory network, Eq. (1) with $J_{ij} = (N-1)^{-1}(\delta_{ij} - 1)$. This matrix has $N-1$ degenerate eigenvalues at $+1/(N-1)$ and a single eigenvalue at $-1$. A similar network configuration (with delays) has been studied as a model of lateral inhibition in the eye of the horseshoe crab, *Limulus* [Coleman and Renninger,1975,1976; Hadeler and Tomiuk,1977; anderHeiden, 1980]. Previous analysis of sustained oscillation in this system has assumed a coherent form for the oscillatory solution, which reduces the problem to a single scalar delay-differential equation. However, by constraining the solution to lie on along the coherent direction, the instability of the oscillatory mode discussed above is not seen. Because of this assumption, fixed-point-only dynamics in the large-gain limit with finite delay are not predicted by previous treatments, to our knowledge.

The behavior of the network at various values of gain and delay are illustrated in Fig.2 for the particular case of N=3. The four regions labeled A,B,C and D characterize the behavior for all N. At low gain ($\beta$ < N-1) the origin is the unique attractor for small delay (region A) and undergoes a Hopf bifurcation at to sustained coherent oscillation at $\tau \sim \pi(\beta^2-1)^{-1/2}$ for large delay (region B). At $\beta$ = N-1 fixed points away from the origin appear. In addition to these fixed points, an oscillatory attractor exists at large gain for $\tau > $ ln [(N–1)/(N–2)]   ( $\cong$ 1/N for large N) (region C). Sustained oscillation does not exist below this critical delay (region D).

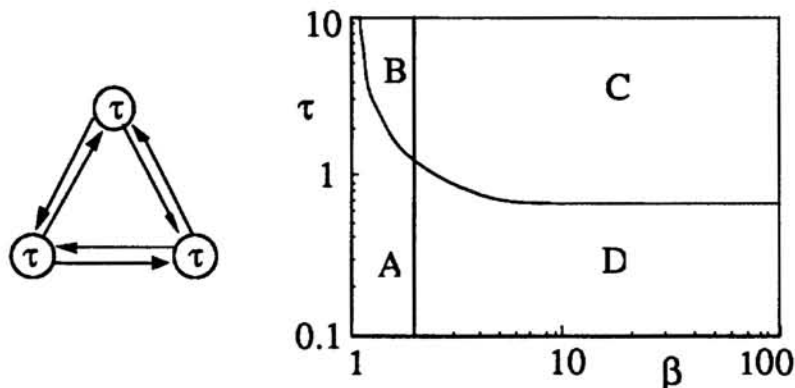

**Figure 2.** Stability Diagram for the All-Inhibitory Delay Network for the Case N = 3. See Text for a Description of A,B,C and D.

As a second example, we consider a ring of delayed neurons. We allow the symmetric connections to be of either sign - that is, connections between neighboring pairs can be mutually excitatory or inhibitory - but are all the same strength. The eigenvalues for the symmetric ring of size N are $\lambda_k$ = cos($2\pi$(k+$\varphi$)/N), where k = 0,1,2...N-1, $\varphi$ = 1/2 if the product of connection strengths around the ring is negative, $\varphi$ = 0 if the product is positive. Borrowing from the language of disordered magnetic systems, a ring which contains an odd number of negative connections (the case $\varphi$ = 1/2) is said to be "frustrated." [Toulouse, 1977]. The large-gain stability analysis for the symmetric ring gives a rather surprising result: *Only frustrated rings with an odd number of neurons will show sustained oscillation.* For this case (N odd and an odd number of negative connections) the critical delay is given by $\tau_{crit}$ = -ln (1 - cos($\pi$/N)). This agrees very well with experimental and numerical data, as does the conclusion that rings with even N do not show sustained oscillation [Marcus and Westervelt, 1989]. The theoretical large-gain critical delay for the all-inhibitory network and the frustrated ring of the same size are compared in Fig. 3. Note that the critical delay for the all-inhibitory network decreases (roughly as 1/N) for larger networks while the ring becomes less prone to oscillation as the network size increases.

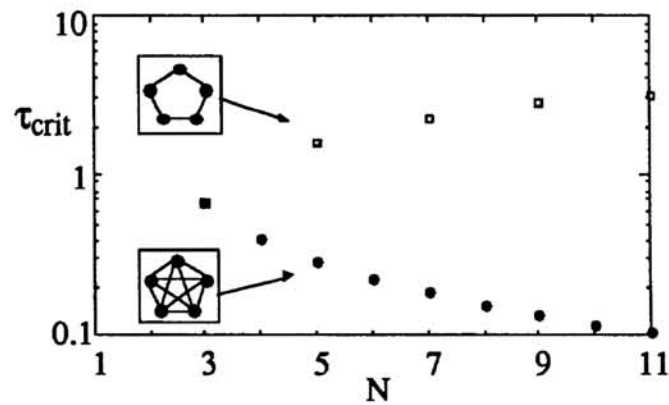

**Figure 3.** Critical Delay from Large-Gain Theory for All-Inhibitory Networks (circles) and Frustrated Rings (squares) of size N.

## III. CHAOS IN NON-SYMMETRIC DELAY NETWORKS

Allowing non-symmetric interconnections greatly expands the repertoire of neural network dynamics and can yield new, powerful computational properties. For example, several recent studies have shown that by using both asymmetric connections and time delay, a neural network can accurately recall of sequences of stored patterns [Kleinfeld,1986; Sompolinsky and Kanter,1986]. It has also been shown that for some parameter values, these pattern-generating networks can produce chaotic dynamics [Riedel, *et al*, 1988].

Relatively little is known about the dynamics of large asymmetric networks [Amari, 1971,1972; Kürten and Clark,1986; Shinomoto,1986; Sompolinsky, *et al*, 1988, Gutfreund,*et al*,1988]. A recent study of continuous-time networks with random asymmetric connections shows that as $N \rightarrow \infty$ these systems will be chaotic whenever the origin is unstable [Sompolinsky,*et al*,1988]. In discrete-state ($\pm 1$) networks,with either parallel or sequential deterministic dynamics, oscillatory modes with long periods are also seen for fully asymmetric random connections ($J_{ij}$ and $J_{ji}$ uncorrelated), but when $J_{ij}$ has either symmetric or antisymmetric correlations short-period attractors seem to predominate [Gutfreund,*et al*,1988]. It is not clear whether the chaotic dynamics of large random networks will appear in small networks with non-symmetric, but non-random, connections.

Small networks with asymmetric connections have been used as models of central pattern generators found in many biological neural systems. [Cohen,*et al*, 1988] These models frequently use time delay to produce sustained rhythmic output, motivated in part by the known presence of time delay in real central pattern generators. General theoretical principles concerning the dynamics of asymmetric network with delay do not exist at present. It has been shown, however, that *large system size is not necessary to produce chaos in neural networks with delay* [e.g. Babcock and Westervelt, 1987]. We find that small systems ($N \geq 3$) with certain asymmetric connections and time delay can produce sustained chaotic oscillation. An example is shown in Fig. 4: These data were produced using an electronic network [Marcus and Westervelt, 1988] of three neurons with

sigmoidal transfer functions $f_1(u(t))=3.8\tanh(8u(t-\tau))$, $f_2(u(t))=2\tanh(6.1u(t))$, $f_3(u(t))=3.5\tanh(2.5u(t))$, connection resistances of $\pm10^5\Omega$ and input capacitances of 10nF. Fig. 4 shows the network configuration and output voltages $V_1$ and $V_2$ for increasing delay in neuron 1. For $\tau < 0.64$ms a periodic attractor similar to the upper left figure is found; for $\tau > 0.97$ms both periodic and chaotic attractors are found.

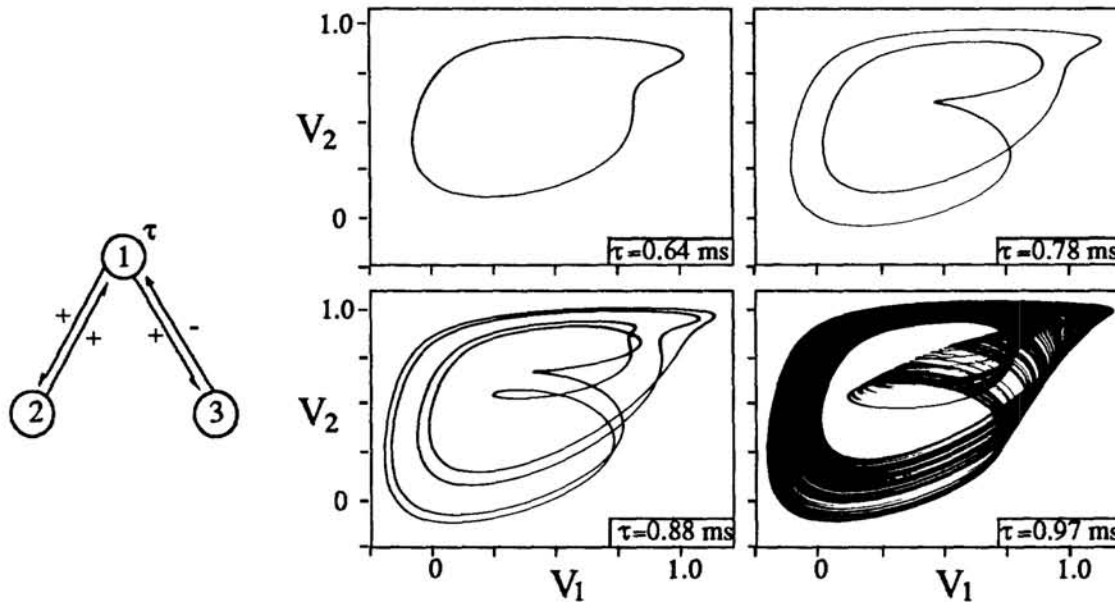

**Figure 4.** Period Doubling to Chaos as the Delay in Neuron 1 is Increased.

Chaos in the network of Fig.4 is closely related to a well-known chaotic delay-differential equation with a noninvertible feedback term [Mackey and Glass,1977]. The noninvertible or "mixed" feedback necessary to produce chaos in the Mackey-Glass equation is achieved in the neural network - which has only monotone transfer functions - by using asymmetric connections.

This association between asymmetry and noninvertible feedback suggests that asymmetric connections may be necessary to produce chaotic dynamics in neural networks, even when time delay is present. This conjecture is further supported by considering the two limiting cases of zero delay and infinite delay, neither of which show chaotic dynamics for symmetric connections.

## IV. CONCLUSION AND OPEN PROBLEMS

We have considered the effects of delayed response in a continuous-time neural network. We find that when the delay of each neuron exceeds a critical value sustained oscillatory modes appear in a symmetric network. Stability analysis yields a design criterion for building stable electronic neural networks, but these results can also be used to created *desired* oscillatory modes in delay networks. For example, a variation of the Hebb rule [Hebb, 1949], created by simply taking the negative of a Hebb matrix, will give negative real eigenvalues corresponding to programed oscillatory patterns. Analyzing the storage capacities and other properties of neural networks with dynamic attractors remain

challenging problems [see, e.g. Gutfreund and Mezard, 1988].

In analyzing the stability of delay systems, we have assumed that the delays and gains of all neurons are identical. This is quite restrictive and is certainly not justified from a biological viewpoint. It would be interesting to study the effects of a wide range of delays in both symmetric and non-symmetric neural networks. It is possible, for example, that the coherent oscillation described above will not persist when the delays are widely distributed.

## Acknowledgements

One of us (CMM) acknowledges support as an AT&T Bell Laboratories Scholar. Research supported in part by JSEP contract N00014-84-K-0465.

## References

S. Amari, 1971, Proc. IEEE, **59**, 35.
S. Amari, 1972, IEEE Trans. **SMC-2**, 643.
U. an der Heiden, 1980, *Analysis of Neural Networks*, Vol. 35 of *Lecture Notes in Biomathematics* (Springer, New York).
K.L. Babcock and R.M. Westervelt, 1987, Physica **28D**, 305.
M.A. Cohen and S. Grossberg, 1983, IEEE Trans. **SMC-13**, 815.
A.H. Cohen, S. Rossignol and S. Grillner, 1988, *Neural Control of Rhythmic Motion*, (Wiley, New York).
B.D. Coleman and G.H. Renninger, 1975, J. Theor. Biol. **51**, 243.
B.D. Coleman and G.H. Renninger, 1976, SIAM J. Appl. Math. **31**, 111.
P. R. Conwell, 1987, in *Proc. of IEEE First Int. Conf. on Neural Networks*,III-95.
H. Gutfreund, J.D. Reger and A.P. Young, 1988, J. Phys. A, **21**, 2775.
H. Gutfreund and M. Mezard, 1988, Phys. Rev. Lett. **61**, 235.
K.P. Hadeler and J. Tomiuk, 1977, Arch. Rat. Mech. Anal. **65**, 87.
D.O. Hebb, 1949, *The Organization of Behavior* (Wiley, New York).
J.J. Hopfield, 1984, Proc. Nat. Acad. Sci. USA **81**, 3008.
D. Kleinfeld, 1984, Proc. Nat. Acad. Sci. USA **83**, 9469.
K.E. Kürten and J.W. Clark, 1986, Phys. Lett. **114A**, 413.
M.C. Mackey and L. Glass, 1977, Science **197**, 287.
C.M. Marcus and R.M. Westervelt, 1988, in: *Proc. IEEE Conf. on Neural Info. Proc. Syst., Denver, CO, 1987*, (American Institute of Physics, New York).
C.M. Marcus and R.M. Westervelt, 1989, Phys. Rev. A **39**, 347.
J.E. Marsden and M. McCracken, *The Hopf Bifurcation and its Applications*, (Springer-Verlag, New York).
U. Riedel, R. Kühn, and J. L. van Hemmen, 1988, Phys. Rev. A **38**, 1105.
S. Shinomoto, 1986, Prog. Theor. Phys. **75**, 1313.
H. Sompolinsky and I. Kanter, 1986, Phys. Rev. Lett. **57**, 259.
H. Sompolinsky, A. Crisanti and H.J. Sommers, 1988, Phys. Rev. Lett. **61**, 259.
G. Toulouse, 1977, Commun. Phys. **2**, 115.